# Model selection in clustering by uniform convergence bounds*

**Joachim M. Buhmann and Marcus Held**
Institut für Informatik III,
Römerstraße 164, D-53117 Bonn, Germany
{jb,held}@cs.uni-bonn.de

## Abstract

Unsupervised learning algorithms are designed to extract structure from data samples. Reliable and robust inference requires a guarantee that extracted structures are typical for the data source, i.e., similar structures have to be inferred from a second sample set of the same data source. The overfitting phenomenon in maximum entropy based annealing algorithms is exemplarily studied for a class of histogram clustering models. Bernstein's inequality for large deviations is used to determine the maximally achievable approximation quality parameterized by a minimal temperature. Monte Carlo simulations support the proposed model selection criterion by finite temperature annealing.

## 1 Introduction

Learning algorithms are designed to extract structure from data. Two classes of algorithms have been widely discussed in the literature – *supervised* and *unsupervised learning*. The distinction between the two classes depends on supervision or teacher information which is either available to the learning algorithm or missing. This paper applies statistical learning theory to the problem of unsupervised learning. In particular, error bounds as a protection against overfitting are derived for the recently developed **Asymmetric Clustering Model** (ACM) for co–occurrence data [6]. These theoretical results show that the continuation method "*deterministic annealing*" yields robustness of the learning results in the sense of statistical learning theory. The computational temperature of annealing algorithms plays the role of a control parameter which regulates the complexity of the learning machine. Let us assume that a hypothesis class $\mathcal{H}$ of loss functions $\mathbf{h}(\mathbf{x}; \alpha)$ is given. These loss functions measure the quality of structures in data. The complexity of $\mathcal{H}$ is controlled by coarsening, i.e., we define a $\gamma$–cover of $\mathcal{H}$. Informally, the inference principle advocated by us performs learning by two inference steps: (i) determine the optimal approximation level $\gamma$ for consistent learning (in terms of large risk deviations); (ii) given the optimal approximation level $\gamma$, average over all hypotheses in an appropriate neighborhood of the empirical minimizer. The result of the inference

*This work has been supported by the German Israel Foundation for Science and Research Development (GIF) under grant #1-0403-001.06/95.

procedure is not a single hypothesis but a set of hypotheses. This set is represented either by an average of loss functions or, alternatively, by a typical member of this set. This induction approach is named *Empirical Risk Approximation* (ERA) [2]. The reader should note that the learning algorithm has to return an average structure which is *typical* in a $\gamma$–cover sense but it is not supposed to return the hypothesis with *minimal empirical risk* as in Vapnik's "Empirical Risk Minimization" (ERM) induction principle for classification and regression [9]. The loss function with minimal empirical risk is usually a structure with maximal complexity, e.g., in clustering the ERM principle will necessarily yield a solution with the maximal number of clusters. The ERM principle, therefore, is not suitable as a model selection principle to determine the number of clusters which are stable under sample fluctuations. The ERA principle with its approximation accuracy $\gamma$ solves this problem by controlling the effective complexity of the hypothesis class.

In spirit, this approach is similar to the Gibbs–algorithm presented for example in [3]. The Gibbs–algorithm samples a random hypothesis from the version space to predict the label of the $l + 1$th data point $x_{l+1}$. The version space is defined as the set of hypotheses which are consistent with the first $l$ given data points. In our approach we use an alternative definition of consistency, where all hypothesis in an appropriate neighborhood of the empirical minimizer define the version space (see also [4]). Averaging over this neighborhood yields a structure with risk equivalent to the expected risk obtained by random sampling from this set of hypotheses. There exists also a tight methodological relationship to [7] and [4] where learning curves for the learning of two class classifiers are derived using techniques from statistical mechanics.

## 2   The Empirical Risk Approximation Principle

The data samples $\mathcal{Z} = \{\mathbf{z}_r \in \Omega, \ 1 \leq r \leq l\}$ which have to be analyzed by the unsupervised learning algorithm are elements of a suitable object (resp. feature) space $\Omega$. The samples are distributed according to a measure $\mu$ which is not assumed to be known for the analysis.[1]

A mathematically precise statement of the ERA principle requires several definitions which formalize the notion of searching for structure in the data. The quality of structures extracted from the data set $\mathcal{Z}$ is evaluated by the *empirical risk* $\hat{\mathcal{R}}(\alpha; \mathcal{Z}) := \frac{1}{l} \sum_{r=1}^{l} \mathbf{h}(\mathbf{z}_r; \alpha)$ of a structure $\alpha$ given the training set $\mathcal{Z}$. The function $\mathbf{h}(\mathbf{z}; \alpha)$ is known as *loss function* in statistics. It measures the costs for processing a generic datum $\mathbf{z}$ with model $\alpha$. Each value $\alpha \in \Lambda$ parameterizes an individual loss function with $\Lambda$ denoting the set of possible parameters. The loss function which minimizes the empirical risk is denoted by $\hat{\alpha}^{\perp} := \arg\min_{\alpha \in \Lambda} \hat{\mathcal{R}}(\alpha; \mathcal{Z})$.

The relevant quality measure for learning is the *expected risk* $\mathcal{R}(\alpha) := \int_{\Omega} \mathbf{h}(\mathbf{z}; \alpha) \, d\mu(\mathbf{z})$. The optimal structure to be inferred from the data is $\alpha^{\perp} := \arg\min_{\alpha \in \Lambda} \mathcal{R}(\alpha)$. The distribution $\mu$ is assumed to decay sufficiently fast with bounded $r$th moments $\mathbf{E}_{\mu} \{|\mathbf{h}(\mathbf{z}; \alpha) - \mathcal{R}(\alpha)|^r\} \leq r! \tau^{r-2} \mathbf{V}_{\mu} \{\mathbf{h}(\mathbf{z}; \alpha)\}, \forall \alpha \in \Lambda$ $(r > 2)$. $\mathbf{E}_{\mu} \{.\}$ and $\mathbf{V}_{\mu} \{.\}$ denote expectation and variance of a random variable, respectively. $\tau$ is a distribution dependent constant.

ERA requires the learning algorithm to determine a set hypotheses on the basis of the finest consistently learnable cover of the hypothesis class. Given a learning accuracy $\gamma$ a subset of parameters $\Lambda_{\gamma} = \{\alpha_1, \ldots, \alpha_{|\Lambda_{\gamma}|-1}\} \cup \{\hat{\alpha}^{\perp}\}$ can be defined such that the hypothesis class $\mathcal{H}$ is covered by the function balls with index sets $\mathcal{B}_{\gamma}(\alpha) := \{\alpha' : \int_{\Omega} |\mathbf{h}(\mathbf{z}; \alpha') - \mathbf{h}(\mathbf{z}; \alpha)| \, d\mu(\mathbf{z}) \leq \gamma\}$, i. e. $\Lambda \subset \bigcup_{\alpha \in \Lambda_{\gamma}} \mathcal{B}_{\gamma}(\alpha)$. The em-

pirical minimizer $\hat{\alpha}^{\perp}$ has been added to the cover to simplify bounding arguments. Large deviation theory is used to determine the approximation accuracy $\gamma$ for learning a hypothesis from the hypothesis class $\mathcal{H}$. The expected risk of the empirical minimizer exceeds the global minimum of the expected risk $\mathcal{R}(\alpha^{\perp})$ by $\epsilon\sigma^{\tau}$ with a probability bounded by Bernstein's inequality [8]

$$
\begin{aligned}
\mathbf{P}\left\{\mathcal{R}(\hat{\alpha}^{\perp}) - \mathcal{R}(\alpha^{\perp}) > \epsilon\sigma^{\tau}\right\} &\leq \mathbf{P}\left\{\sup_{\alpha\in\Lambda_{\gamma}}|\hat{\mathcal{R}}(\alpha) - \mathcal{R}(\alpha)| \geq \frac{1}{2}\left(\epsilon\sigma^{\tau} - \gamma\right)\right\} \\
&\leq 2|\Lambda_{\gamma}|\exp\left(-\frac{l\left(\epsilon - \gamma/\sigma^{\tau}\right)^2}{8 + 4\tau\left(\epsilon - \gamma/\sigma^{\tau}\right)}\right) \equiv \delta. \quad (1)
\end{aligned}
$$

The complexity $|\Lambda_{\gamma}|$ of the coarsened hypothesis class has to be small enough to guarantee with high confidence small $\epsilon$–deviations.[2] This large deviation inequality weighs two competing effects in the learning problem, i. e. the probability of a large deviation exponentially decreases with growing sample size $l$, whereas a large deviation becomes increasingly likely with growing cardinality of the $\gamma$–cover of the hypothesis class. According to (1) the sample complexity $l_0(\gamma, \epsilon, \delta)$ is defined by

$$
\log|\Lambda_{\gamma}| - \frac{l_0\left(\epsilon - \gamma/\sigma^{\tau}\right)^2}{8 + 4\tau\left(\epsilon - \gamma/\sigma^{\tau}\right)} + \log\frac{2}{\delta} = 0. \quad (2)
$$

With probability $1 - \delta$ the deviation of the empirical risk from the expected risk is bounded by $\frac{1}{2}\left(\epsilon^{\mathrm{opt}}\sigma^{\tau} - \gamma\right) =: \gamma^{\mathrm{app}}$. Averaging over a set of functions which exceed the empirical minimizer by no more than $2\gamma^{\mathrm{app}}$ in empirical risk yields an average hypothesis corresponding to the statistically significant structure in the data, i.e., $\hat{\mathcal{R}}(\alpha^{\perp}) - \hat{\mathcal{R}}(\hat{\alpha}^{\perp}) \leq \mathcal{R}(\alpha^{\perp}) + \gamma^{\mathrm{app}} - (\mathcal{R}(\hat{\alpha}^{\perp}) - \gamma^{\mathrm{app}}) \leq 2\gamma^{\mathrm{app}}$ since $\mathcal{R}(\alpha^{\perp}) \leq \mathcal{R}(\hat{\alpha}^{\perp})$ by definition. The key task in the following remains to calculate the minimal precision $\epsilon(\gamma)$ as a function of the approximation $\gamma$ and to bound from above the cardinality $|\Lambda_{\gamma}|$ of the $\gamma$–cover for specific learning problems.

## 3  Asymmetric clustering model

The asymmetric clustering model was developed for the analysis resp. grouping of objects characterized by co–occurrence of objects and certain feature values [6]. Application domains for this explorative data analysis approach are for example texture segmentation, statistical language modeling or document retrieval.

Denote by $\Omega = \mathcal{X} \times \mathcal{Y}$ the product space of objects $\mathbf{x}_i \in \mathcal{X}, 1 \leq i \leq n$ and features $\mathbf{y}_j \in \mathcal{Y}, 1 \leq j \leq f$. The $\mathbf{x}_i \in \mathcal{X}$ are characterized by observations $\mathcal{Z} = \{\mathbf{z}_r\} = \{(\mathbf{x}_{i(r)}, \mathbf{y}_{j(r)}), r = 1, \ldots, l\}$. The sufficient statistics of how often the object–feature pair $(\mathbf{x}_i, \mathbf{y}_j)$ occurs in the data set $\mathcal{Z}$ is measured by the set of frequencies $\{\eta_{ij} : \text{number of observations} (\mathbf{x}_i, \mathbf{y}_j) /\text{total number of observations}\}$. Derived measurements are the frequency of observing object $\mathbf{x}_i$, i. e. $\eta_i = \sum_{j=1}^{f} \eta_{ij}$ and the frequency of observing feature $\mathbf{y}_j$ given object $\mathbf{x}_i$, i. e. $\eta_{j|i} = \eta_{ij}/\eta_i$. The asymmetric clustering model defines a generative model of a finite mixture of component probability distributions in feature space with cluster–conditional distributions $\mathbf{q} = (q_{j|\nu}), 1 \leq j \leq f, 1 \leq \nu \leq k$ (see [6]). We introduce indicator variables $\mathbf{M}_{i\nu} \in \{0, 1\}$ for the membership of object $\mathbf{x}_i$ in cluster $\nu \in \{1, \ldots, k\}$. $\sum_{\nu=1}^{k} \mathbf{M}_{i\nu} = 1 \ \forall i : 1 \leq i \leq n$ enforces the uniqueness constraint for assignments.

Using these variables the observed data $\mathcal{Z}$ are distributed according to the generative model over $\mathcal{X} \times \mathcal{Y}$:

$$\mathbf{P}\left\{\mathbf{x}_i, \mathbf{y}_j | \mathbf{M}, \mathbf{q}\right\} = \frac{1}{n} \sum_{\nu=1}^{k} \mathbf{M}_{i\nu} q_{j|\nu}. \qquad (3)$$

For the analysis of the unknown data source — characterized (at least approximatively) by the empirical data $\mathcal{Z}$ — a structure $\alpha = (\mathbf{M}, \mathbf{q})$ with $\mathbf{M} \in \{0, 1\}^{n \times k}$ has to be inferred. The aim of an ACM analysis is to group the objects $\mathbf{x}_i$ as coded by the unknown indicator variables $\mathbf{M}_{i\nu}$ and to estimate for each cluster $\nu$ a prototypical feature distribution $q_{j|\nu}$.

Using the *loss function* $\mathbf{h}(\mathbf{x}_i, \mathbf{y}_j; \alpha) = \log n - \sum_{\nu=1}^{k} \mathbf{M}_{i\nu} \log q_{j|\nu}$ the maximization of the likelihood can be formulated as minimization of the *empirical risk*: $\hat{\mathcal{R}}(\alpha; \mathcal{Z}) = \sum_{i=1}^{n} \sum_{j=1}^{f} \eta_{ij} \mathbf{h}(\mathbf{x}_i, \mathbf{y}_j; \alpha)$, where the essential quantity to be minimized is the *expected risk*: $\mathcal{R}(\alpha) = \sum_{i=1}^{n} \sum_{j=1}^{f} \mathbf{P}^{\text{true}}\{\mathbf{x}_i, \mathbf{y}_j\} \mathbf{h}(\mathbf{x}_i, \mathbf{y}_j; \alpha)$. Using the maximum entropy principle the following annealing equations are derived [6]:

$$\hat{q}_{j|\nu} = \frac{\sum_{i=1}^{n} \langle \mathbf{M}_{i\nu} \rangle \eta_{ij}}{\sum_{i=1}^{n} \langle \mathbf{M}_{i\nu} \rangle} = \sum_{i=1}^{n} \frac{\langle \mathbf{M}_{i\nu} \rangle \eta_i}{\sum_{h=1}^{n} \langle \mathbf{M}_{h\nu} \rangle} \eta_{j|i}, \qquad (4)$$

$$\langle \mathbf{M}_{i\nu} \rangle = \frac{\exp\left[\beta \sum_{j=1}^{f} \eta_{j|i} \log q_{j|\nu}\right]}{\sum_{\mu=1}^{k} \exp\left[\beta \sum_{j=1}^{f} \eta_{j|i} \log q_{j|\mu}\right]}. \qquad (5)$$

**The critical temperature:** Due to the limited precision of the observed data it is natural to study histogram clustering as a learning problem with the hypothesis class $\mathcal{H} = \{-\sum_{\nu} \mathbf{M}_{i\nu} \log q_{j|\nu} : \mathbf{M}_{i\nu} \in \{0, 1\} \wedge \sum_{\nu} \mathbf{M}_{i\nu} = 1 \wedge q_{j|\nu} \in \{\frac{1}{l}, \frac{2}{l}, \cdots, 1\} \wedge \sum_{j} q_{j|\nu} = 1\}$. The limited number of observations results in a limited precision of the frequencies $\eta_{j|i}$. The value $q_{j|\nu} = 0$ has been excluded since it causes infinite expected risk for $\mathbf{P}^{\text{true}}\{\mathbf{y}_j | \mathbf{x}_i\} > 0$. The size of the regularized hypothesis class $\Lambda_{\gamma}$ can be upper bounded by the cardinality of the complete hypothesis class divided by the minimal cardinality of a $\gamma$–function ball centered at a function of the $\gamma$–cover $\Lambda_{\gamma}$, i. e. $|\Lambda_{\gamma}| \leq |\mathcal{H}| \big/ \min_{\tilde{\alpha} \in \Lambda_{\gamma}} |\mathcal{B}_{\gamma}(\tilde{\alpha})|$.

The cardinality of a function ball with radius $\gamma$ can be approximated by adopting techniques from asymptotic analysis [1] ($\Theta(x) = \left\{\begin{smallmatrix} 1 \\ 0 \end{smallmatrix}\right.$ for $x \gtrless 0$):

$$|\mathcal{B}_{\gamma}(\tilde{\alpha})| = \sum_{\mathcal{M}} \sum_{\{q_{j|\alpha}\}} \Theta\left(\gamma - \sum_{i,j} \frac{1}{n} \mathbf{P}^{\text{true}}\{\mathbf{y}_j | \mathbf{x}_i\} \left|\log \frac{q_{j|m(i)}}{\tilde{q}_{j|\tilde{m}(i)}}\right|\right) \qquad (6)$$

$$= \frac{k^n l^{kf}}{(2\pi\imath)^k} \int_0^1 \cdots \int_0^1 \prod_{j=1}^{f} \prod_{\nu=1}^{k} dq_{j|\nu} \int_{-\imath\infty}^{+\imath\infty} \cdots \int_{-\imath\infty}^{+\imath\infty} dQ_{\nu} \int_{-\imath\infty}^{+\imath\infty} \frac{d\hat{x}}{2\pi\imath\hat{x}} \exp\left(n\mathcal{S}(\mathbf{q}, \mathbf{Q}, \hat{x})\right),$$

and the entropy $\mathcal{S}$ is given by

$$\mathcal{S}(\mathbf{q}, \mathbf{Q}, \hat{x}) = \gamma\hat{x} - \sum_{\nu} Q_{\nu}\left(\sum_{j} q_{j|\nu} - 1\right) +$$
$$\frac{1}{n} \sum_{i} \log \sum_{\rho} \exp\left(-\hat{x} \sum_{j} \mathbf{P}^{\text{true}}\{\mathbf{y}_j | \mathbf{x}_i\} \left|\log \frac{q_{j|\rho}}{\tilde{q}_{j|\tilde{m}(i)}}\right|\right). \qquad (7)$$

The auxiliary variables $\mathbf{Q} = \{Q_{\nu}\}_{\nu=1}^{k}$ are Lagrange parameters to enforce the normalizations $\sum_{j} q_{j|\nu} = 1$. Choosing $q_{j|\alpha} = \tilde{q}_{j|\tilde{m}(i)} \forall \tilde{m}(i) = \alpha$, we obtain an approximation of the integral. The reader should note that a saddlepoint approximation in

the usual sense is only applicable for the parameter $\hat{x}$ but will fail for the $\mathbf{q}, \mathbf{Q}$ parameters since the integrand is maximal at the non-differentiability point of the absolute value function. We, therefore, expand $\mathcal{S}(\mathbf{q}, \mathbf{Q}, \hat{x})$ up to linear terms $\mathcal{O}(\mathbf{q} - \tilde{\mathbf{q}})$ and integrate piece-wise.

Using the abbreviation $\kappa_{i\nu} := \sum_j \mathbf{P}^{\text{true}} \{\mathbf{y}_j | \mathbf{x}_i\} \left| \log \frac{\tilde{q}_{j|\nu}}{\tilde{q}_{j|\hat{m}(i)}} \right|$ the following saddle point approximation for the integral over $\hat{x}$ is obtained:

$$\gamma = \frac{1}{n} \sum_{i=1}^n \sum_{\mu=1}^k \mathbf{P}_{i\mu} \kappa_{j|\mu} \quad \text{with} \quad \mathbf{P}_{i\alpha} = \frac{\exp(-\hat{x}\kappa_{i\alpha})}{\sum_\mu \exp(-\hat{x}\kappa_{i\mu})}. \tag{8}$$

The entropy $\mathcal{S}$ evaluated at $\mathbf{q} = \tilde{\mathbf{q}}$ yields in combination with the Laplace approximation [1] an estimate for the cardinality of the $\gamma$–cover

$$\log |\Lambda_\gamma| = n(\log k - \mathcal{S}) + \frac{1}{2} \sum_{i,\rho} \kappa_{i\rho} \mathbf{P}_{i\rho} \left( \sum_\nu \mathbf{P}_{i\nu} \kappa_{i\nu} - \kappa_{i\rho} \right) \hat{x}^2 \tag{9}$$

where the second term results from the second order term of the Taylor expansion around the saddle point. Inserting this complexity in equation (2) yields an equation which determines the required number of samples $l_0$ for a fixed precision $\epsilon$ and confidence $\delta$. This equation defines a functional relationship between the precision $\epsilon$ and the approximation quality $\gamma$ for fixed sample size $l_0$ and confidence $\delta$. Under this assumption the precision $\epsilon$ depends on $\gamma$ in a non–monotone fashion, i. e.

$$\epsilon = \frac{\gamma}{\sigma^\tau} + \frac{2}{l_0} \left[ \sqrt{2l_0 C + \tau^2 C^2} + \tau C \right], \tag{10}$$

using the abbreviation $C = \log |\Lambda_\gamma| + \log \frac{2}{\delta}$. The minimum of the function $\epsilon(\gamma)$ defines a compromise between uncertainty originating from empirical fluctuations and the loss of precision due to the approximation by a $\gamma$–cover. Differentiating with respect to $\gamma$ and setting the result to zero $(d\epsilon(\gamma)/d\gamma = 0)$ yields as upper bound for the inverse temperature:

$$\hat{x} \leq \frac{1}{\sigma^\tau} \frac{l_0}{2n} \left( \tau + \frac{l_0 + C\tau^2}{\sqrt{2l_0 C + \tau^2 C^2}} \right)^{-1}. \tag{11}$$

Analogous to estimates of $k$–means, phase–transitions occur in ACM while lowering the temperature. The mixture model for the data at hand can be partitioned into more and more components, revealing finer and finer details of the generation process. The critical $\hat{x}^{\text{opt}}$ defines the resolution limit below which details can not be resolved in a reliable fashion on the basis of the sample size $l_0$.

Given the inverse temperature $\hat{x}$ the effective cardinality of the hypothesis class can be upper bounded via the solution of the fix point equation (8). On the other hand this cardinality defines with (11) and the sample size $l_0$ an upper bound on $\hat{x}$. Iterating these two steps we finally obtain an upper bound for the critical inverse temperature given a sample size $l_0$.

**Empirical Results:**
For the evaluation of the derived theoretical result a series of Monte–Carlo experiments on artificial data has been performed for the asymmetric clustering model. Given the number of objects $n = 30$, the number of groups $k = 5$ and the size of the histograms $f = 15$ the generative model for this experiments was created randomly and is summarized in fig. 1. From this generative model sample sets of arbitrary size can be generated and the true distributions $\mathbf{P}^{\text{true}} \{\mathbf{y}_j | \mathbf{x}_i\}$ can be calculated. In figure 2a,b the predicted temperatures are compared to the empirically observed critical temperatures, which have been estimated on the basis of 2000 different samples of randomly generated co–occurrence data for each $l_0$. The expected risk (solid)

| $\nu$ | $q_{j\vert\nu}$ |
|---|---|
| 1 | $\{0.11, 0.01, 0.11, 0.07, 0.08, 0.04, 0.06, 0, 0.13, 0.07, 0.08, 0.1, 0, 0.11, 0.03\}$ |
| 2 | $\{0.18, 0.1, 0.09, 0.02, 0.05, 0.09, 0.08, 0.03, 0.06, 0.07, 0.03, 0.02, 0.07, 0.06, 0.05\}$ |
| 3 | $\{0.17, 0.05, 0.05, 0.06, 0.06, 0.05, 0.03, 0.11, 0.09, 0, 0.02, 0.1, 0.03, 0.07, 0.11\}$ |
| 4 | $\{0.15, 0.07, 0.1, 0.03, 0.09, 0.03, 0.04, 0.05, 0.06, 0.05, 0.08, 0.04, 0.08, 0.09, 0.04\}$ |
| 5 | $\{0.09, 0.09, 0.07, 0.1, 0.07, 0.06, 0.06, 0.11, 0.07, 0.07, 0.1, 0.02, 0.07, 0.02, 0\}$ |

$m(i) = (5, 3, 2, 5, 2, 2, 5, 4, 2, 2, 2, 4, 1, 5, 3, 5, 3, 4, 1, 2, 2, 3, 1, 1, 2, 5, 5, 2, 2, 1)$

Figure 1: Generative ACM model for the Monte–Carlo experiments.

and empirical risk (dashed) of these 2000 inferred models are averaged. Overfitting sets in when the expected risk rises as a function of the inverse temperature $\hat{x}$.

Figure 2c indicates that on average the minimal expected risk is assumed when the effective number is smaller than or equal 5, i. e. the number of clusters of the true generative model. Predicting the right computational temperature, therefore, also enables the data analyst to solve the cluster validation problem for the asymmetric clustering model. Especially for $l_0 = 800$ the sample fluctuations do not permit the estimate of five clusters and the minimal computational temperature prevents such an inference result. On the other hand for $l_0 = 1600$ and $l_0 = 2000$ the minimal temperature prevents the algorithm to infer too many clusters, which would be an instance of overfitting.

As an interesting point one should note that for an infinite number of observations the critical inverse temperature reaches a finite positive value and not more than the five effective clusters are extracted. At this point we conclude, that for the case of histogram clustering the *Empirical Risk Approximation* solves for realizable rules the problem of model validation, i. e. choosing the right number of clusters.

Figure 2d summarizes predictions of the critical temperature on the basis of the empirical distribution $\eta_{ij}$ rather than the true distribution $\mathbf{P}^{\mathrm{true}}\{\mathbf{x}_i, \mathbf{y}_j\}$. The empirical distribution has been generated by a training sample set with $\hat{x}$ of eq. (11) being used as a plug-in estimator. The histogram depicts the predicted inverse temperature for $l_0 = 1200$. The average of these plug-in estimators is equal to the predicted temperature for the true distribution. The estimates of $\hat{x}$ are biased towards too small inverse temperatures due to correlations between the parameter estimates and the stopping criterion. It is still an open question and focus of ongoing work to rigorously bound the variance of this plug-in estimator.

Empirically we observe a reduction of the variance of the expected risk occurring at the predicted temperature for higher sample sizes $l_0$.

## 4 Conclusions

The two conditions that the empirical risk has to uniformly converge towards the expected risk and that all loss functions within an $2\gamma^{\mathrm{app}}$-range of the global empirical risk minimum have to be considered in the inference process limits the complexity of the underlying hypothesis class for a given number of samples. The maximum entropy method which has been widely employed in deterministic annealing procedures for optimization problems is substantiated by our analysis. Solutions with too many clusters clearly overfit the data and do not generalize. The condition that the hypothesis class should only be divided in function balls of size $\gamma$ forces us to stop the stochastic search at the lower bound of the computational temperature.

Another important result of this investigation is the fact that choosing the right stopping temperature for the annealing process not only avoids overfitting but also solves the cluster validation problem in the realizable case of ACM. A possible inference of too many clusters using the empirical risk functional is suppressed.

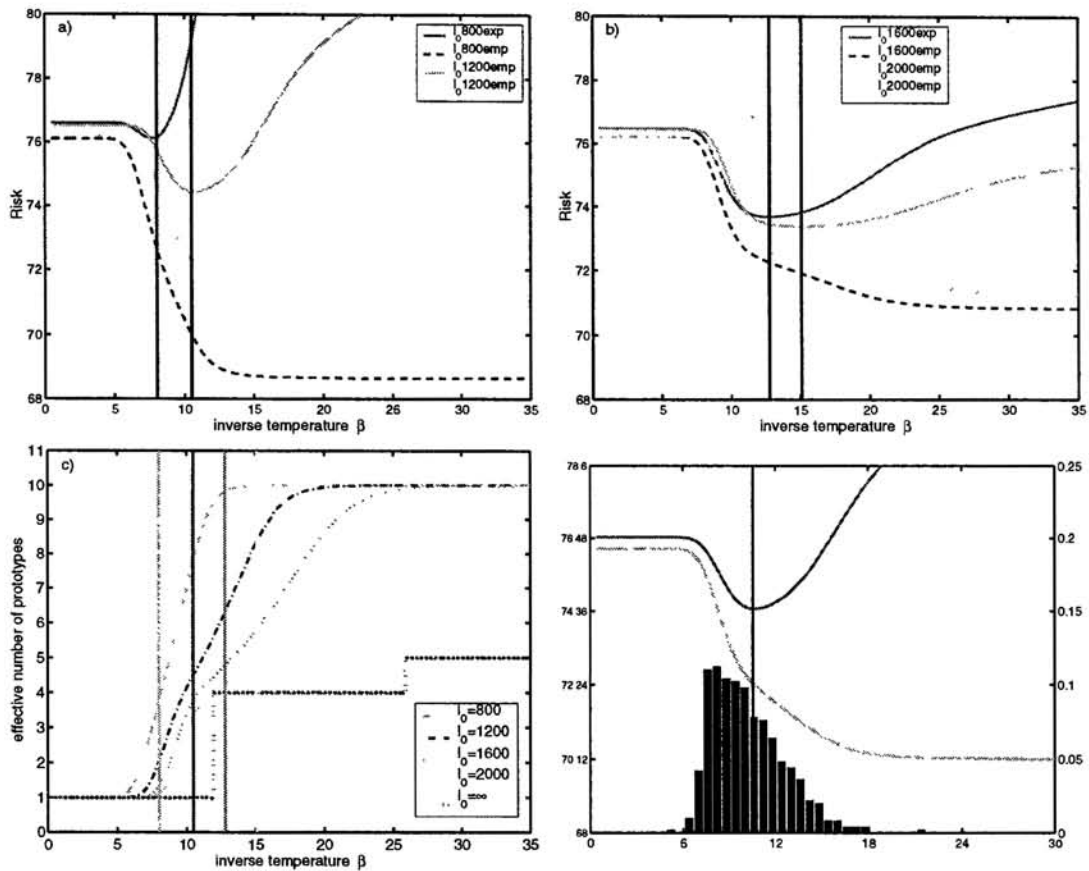

Figure 2: Comparison between the theoretically derived upper bound on $\hat{x}$ and the observed critical temperatures (minimum of the expected risk vs. $\hat{x}$ curve). Depicted are the plots for $l_0 = 800, 1200, 1600, 2000$. Vertical lines indicate the predicted critical temperatures. The average effective number of clusters is drawn in part c. In part d the distribution of the plug–in estimates is shown for $l_0 = 1200$.

## Footnotes

[1]Knowledge of covering numbers is required in the following analysis which is a weaker type of information than complete knowledge of the probability measure $\mu$ (see also [5]).

[2]The maximal standard deviation $\sigma^{\tau} := \sup_{\alpha\in\Lambda_{\gamma}} \sqrt{\mathbf{V}\{\mathbf{h}(\mathbf{z}; \alpha)\}}$ defines the scale to measure deviations of the empirical risk from the expected risk (see [2]).

# References

[1] N. G. De Bruijn. *Asymptotic Methods in Analysis.* North-Holland Publishing Co., (repr. Dover), Amsterdam, 1958, (1981).

[2] J. M. Buhmann. Empirical risk approximation. Technical Report IAI-TR 98-3, Institut für Informatik III, Universität Bonn, 1998.

[3] D. Haussler, M. Kearns, and R. Schapire. Bounds on the sample complexity of Bayesian learning using information theory and the VC dimension. *Machine Learning,* 14(1):83–113, 1994.

[4] D. Haussler, M. Kearns, H.S. Seung, and N. Tishby. Rigorous learning curve bounds from statistical mechanics. *Machine Learning,* 25:195–236, 1997.

[5] D. Haussler and M. Opper. Mutual information, metric entropy and cumulative relative entropy risk. *Annals of Statistics,* December 1996.

[6] T. Hofmann, J. Puzicha, and M.I. Jordan. Learning from dyadic data. In M. S. Kearns, S. A. Solla, and D. A. Cohn, editors, *Advances in Neural Information Processing Systems 11.* MIT Press, 1999. to appear.

[7] H. S. Seung, H. Sompolinsky, and N. Tishby. Statistical mechanics of learning from examples. *Physical Review A,* 45(8):6056–6091, April 1992.

[8] A. W. van der Vaart and J. A. Wellner. *Weak Convergence and Empirical Processes.* Springer-Verlag, New York, Berlin, Heidelberg, 1996.

[9] V. N. Vapnik. *Statistical Learning Theory.* Wiley–Interscience, New York, 1998.